# A COMPUTER SIMULATION OF CEREBRAL NEOCORTEX:
## COMPUTATIONAL CAPABILITIES OF NONLINEAR NEURAL NETWORKS

Alexander Singer* and John P. Donoghue**

*Department of Biophysics, Johns Hopkins University, Baltimore, MD 21218 (to whom all correspondence should be addressed)

**Center for Neural Science, Brown University, Providence, RI 02912

A synthetic neural network simulation of cerebral neocortex was developed based on detailed anatomy and physiology. Processing elements possess temporal nonlinearities and connection patterns similar to those of cortical neurons. The network was able to replicate spatial and temporal integration properties found experimentally in neocortex. A certain level of randomness was found to be crucial for the robustness of at least some of the network's computational capabilities. Emphasis was placed on how synthetic simulations can be of use to the study of both artificial and biological neural networks.

A variety of fields have benefited from the use of computer simulations. This is true in spite of the fact that general theories and conceptual models are lacking in many fields and contrasts with the use of simulations to explore existing theoretical structures that are extremely complex (cf. MacGregor and Lewis, 1977). When theoretical superstructures are missing, simulations can be used to synthesize empirical findings into a system which can then be studied analytically in and of itself. The vast compendium of neuroanatomical and neurophysiological data that has been collected and the concomitant absence of theories of brain function (Crick, 1979; Lewin, 1982) makes neuroscience an ideal candidate for the application of synthetic simulations. Furthermore, in keeping with the spirit of this meeting, neural network simulations which synthesize biological data can make contributions to the study of artificial neural systems as general information processing machines as well as to the study of the brain. A synthetic simulation of cerebral neocortex is presented here and is intended to be an example of how traffic might flow on the two-way street which this conference is trying to build between artificial neural network modelers and neuroscientists.

The fact that cerebral neocortex is involved in some of the highest forms of information processing and the fact that a wide variety of neurophysiological and neuroanatomical data are amenable to simulation motivated the present development of a synthetic simulation of neocortex. The simulation itself is comparatively simple; nevertheless it is more realistic in terms of its structure and elemental processing units than most artificial neural networks.

The neurons from which our simulation is constructed go beyond the simple sigmoid or hard-saturation nonlinearities of most artificial neural systems. For example,

because inputs to actual neurons are mediated by ion currents whose driving force depends on the membrane potential of the neuron, the amplitude of a cell's response to an input, i.e. the amplitude of the post-synaptic potential (PSP), depends not only on the strength of the synapse at which the input arrives, but also on the state of the neuron at the time of the input's arrival. This aspect of classical neuron electrophysiology has been implemented in our simulation (figure 1A), and leads to another important nonlinearity of neurons: namely, current shunting. Primarily effective as shunting inhibition, excitatory current can be shunted out an inhibitory synapse so that the sum of an inhibitory postsynaptic potential and an excitatory postsynaptic potential of equal amplitude does not result in mutual cancellation. Instead, interactions between the ion reversal potentials, conductance values, relative timing of inputs, and spatial locations of synapses determine the amplitude of the response in a nonlinear fashion (figure 1B) (see Koch, Poggio, and Torre, 1983 for a quantitative analysis). These properties of actual neurons have been ignored by most artificial neural network designers, though detailed knowledge of them has existed for decades and in spite of the fact that they can be used to implement complex computations (e.g. Torre and Poggio, 1978; Houchin, 1975).

The development of action potentials and spatial interactions within the model neurons have been simplified in our simulation. Action potentials involve preprogrammed fluctuations in the membrane potential of our neurons and result in an absolute and a relative refractory period. Thus, during the time a cell is firing a spike synaptic inputs are ignored, and immediately following an action potential the neuron is hyperpolarized. The modeling of spatial interactions is also limited since neurons are modeled primarily as spheres. Though the spheres can be deformed through control of a synaptic weight which modulates the amplitudes of ion conductances, detailed dendritic interactions are not simulated. Nonetheless, the fact that inhibition is generally closer to a cortical neuron's soma while excitation is more distal in a cell's dendritic tree is simulated through the use of stronger inhibitory synapses and relatively weaker excitatory synapses.

The relative strengths of synapses in a neural network define its connectivity. Though initial connectivity is random in many artificial networks, brains can be thought to contain a combination of randomness and fixed structure at distinct levels (Szentagothai, 1978). From a macroscopic perspective, all of cerebral neocortex might be structured in a modular fashion analogous to the way the barrel field of mouse somatosensory cortex is structured (Woolsey and Van der Loos, 1970). Though speculative, arguments for the existence of some sort of anatomical modularity over the entire cortex are gaining ground

(Mountcastle, 1978; Szentagothai, 1979; Shepherd, in press). Thus, inspired by the barrels of mice and by growing interest in functional units of 50 to 100 microns with on the order of 1000 neurons, our simulation is built up of five modules (60 cells each) with more dense local interconnections and fewer intermodular contacts. Furthermore, a wide variety of neuronal classification schemes have led us to subdivide the gross structure of each module so as to contain four classes of neurons: cortico-cortical pyramids, output pyramids, spiny stellate or local excitatory cells, and GABAergic or inhibirtory cells.

At this level of analysis, the impressed structure allows for control over a variety of pathways. In our simulation each class of neurons within a module is connected to every other class and intermodular connections are provided along pathways from cortico-cortical pyramids to inhibitory cells, output pyramids, and cortico-cortical pyramids in immediately adjacent modules. A general sense of how strong a pathway is can be inferred from the product of the number of synapses a neuron receives from a particular class and the strength of each of those synapses. The broad architecture of the simulation is further structured to emphasize a three step path: Inputs to the network impact most strongly on the spiny stellate cells of the module receiving the input; these cells in turn project to cortico-cortical pyramidal cells more strongly than they do to other cell types; and finally, the pathway from the cortico-cortical pyramids to the output pyramidal cells of the same module is also particularly strong. This general architecture (figure 2) has received empirical support in many regions of cortex (Jones, 1986).

In distinction to this synaptic architecture, a fine-grain connectivity is defined in our simulated network as well. At a more microscopic level, connectivity in the network is random. Thus, within the confines of the architecture described above, the determination of which neuron of a particular class is connected to which other cell in a target class is done at random. Two distinct levels of connectivity have, therefore, been established (figure 3). Together they provide a middle ground between the completely arbitrary connectivity of many artificial neural networks and the problem specific connectivities of other artificial systems. This distinction between gross synaptic architecture and fine-grain connectivity also has intuitive appeal for theories of brain development and, as we shall see, has non-trivial effects on the computational capabilities of the network as a whole.

With defintions for input integration within the local processors, that is within the neurons, and with the establishment of connectivity patterns, the network is complete and ready to perform as a computational unit. In order to judge the simulation's capabilities in some rough way, a qualitative analysis of its response to an input will suffice. Figure 4

shows the response of the network to an input composed of a small burst of action potentials arriving at a single module. The data is displayed as a raster in which time is mapped along the abscissa and all the cells of the network are arranged by module and cell class along the ordinate. Each marker on the graph represents a single action potential fired by the appropriate neuron at the indicated time. Qualitatively, what is of importance is the fact that the network does not remain unresponsive, saturate with activity in all neurons, or oscillate in any way. Of course, that the network behave this way was predetermined by the combination of the properties of the neurons with a judicious selection of synaptic weights and path strengths. The properties of the neurons were fixed from physiological data, and once a synaptic architecture was found which produced the results in figure 4, that too was fixed. A more detailed analysis of the temporal firing pattern and of the distribution of activity over the different cell classes might reveal important network properties and the relative importance of various pathways to the overall function. Such an analysis of the sensitivity of the network to different path strengths and even to intracellular parameters will, however, have to be postponed. Suffice it to say at this point that the network, as structured, has some nonzero, finite, non-oscillatory response which, qualitatively, might not offend a physiologist judging cortical activity.

Though the synaptic architecture was tailored manually and fixed so as to produce "reasonable" results, the fine-grain connectivity , i.e. the determination of exactly which cell in a class connects to which other cell, was random. An important property of artificial (and presumably biological) neural networks can be uncovered by exploiting the distinction between levels of connectivity described above. Before doing so, however, a detail of neural network design must be made explicit. Any network, either artificial or biological, must contend with the time it takes to communicate among the processing elements. In the brain, the time it takes for an action potential to travel from one neuron to another depends on the conduction velocity of the axon down which the spike is traveling and on the delay that occurs at the synapse connecting the cells. Roughly, the total transmission time from one cortical neuron to another lies between 1 and 5 milliseconds. In our simulation two

paradigms were used. In one case, the transmission times between all neurons were standardized at 1 msec.* Alternatively, the transmission times were fixed at random, though admittedly unphysiological, values between 0.1 and 2 msec.

Now, if the time it takes for an action potential to travel from one neuron to another were fixed for all cells at 1 msec, different fine-grain connectivity patterns are found to produce entirely distinct network responses to the same input, in spite of the fact that the gross synaptic architecture remained constant. This was true no matter what particular synaptic architecture was used. If, on the other hand, one changes the transmission times so that they vary randomly between 0.1 and 2 msec, it becomes easy to find sets of synaptic strengths that were robust with respect to changes in the fine-grain connectivity. Thus, a wide search of path strengths failed to produce a network which was robust to changes in fine-grain connectivity in the case of identical transmission times, while a set of synaptic weights that produced robust responses was easy to find when the transmission times were randomized. Figure 5 summarizes this result. In the figure overall network activity is measured simply as the total number of action potentials generated by pyramidal cells during an experiment and robustness can be judged as the relative stability of this response. The abscissa plots distinct experiments using the same synaptic architecture with different fine-grain connectivity patterns. Thus, though the synaptic architecture remains constant, the different trials represent changes in which particular cell is connected to which other cell. The results show quite dramatically that the network in which the transmission times are randomly distributed is more robust with respect to changes in fine-grain connectivity than the network in which the transmission times are all 1 msec.

It is important to note that in either case, both when the network was robust and when changes of fine-grain connectivity produced gross changes in network output, the synaptic architectures produced outputs like that in figure 4 with some fine-grain connectivities. If the response of the network to an input can be considered the result of

some computation, figure 5 reveals that the *same* computational capability is not robust with respect to changes in fine-grain connectivity when transmission times between neurons are all 1 msec, but is more robust when these times are randomized. Thus, a single computational capability, viz. a response like that in figure 4 to a single input, was found to exist in networks with different synaptic architectures and different transmission time paradigms; this computational capability, however, varied in terms of its robustness with respect to changes in fine-grain connectivity when present in either of the transmission time paradigms.

A more complex computational capability emerged from the neural network simulation we have developed and described. If we label two neighboring modules C2 and C3, an input to C2 will suppress the response of C3 to a second input at C3 if the second input is delayed. A convenient way of representing this spatio-temporal integration property is given in figure 6. The ordinate plots the ratio of the normal response of one module (say C3) to the response of the module to the same input when an input to a neighboring module (say C2) preceeds the input to the original module (C3). Thus, a value of one on the ordinate means the earlier spatially distinct input had no effect on the response of the module in which this property is being measured. A value less than one represents suppression, while values greater than one represent enhancement. On the abscissa, the interstimulus interval is plotted. From figure 6, it can be seen that significant suppression of the pyramidal cell output, mostly of the output pyramidal cell output, occurs when the inputs are separated by 10 to 30 msec. This response can be characterized as a sort of dynamic lateral inhibition since an input is suppressing the ability of a neighboring region to respond when the input pairs have a particular time course. This property could play a variety of role in biological and artificial neural networks. One role for this spatio-temporal integration property, for example, might be in detecting the velocity of a moving stimulus.

The emergent spatio-temporal property of the network just described was not explicitly built into the network. Moreover, no set of synaptic weights was able to give rise to this computational capability when transmission times were all set to 1 msec. Thus, in addition to providing robustness, the random transmission times also enabled a more complex property to emerge. The important factor in the appearances of both the robustness and the dynamic lateral inhibition was randomization; though it was implemented as randomly varying transmission times, random spontaneous activity would have played the same role. From the viewpoint, then, of the engineer designing artificial neural networks, the neural network presented here has instructional value in spite of the

fact that it was designed to synthesize biological data. Specifically, it motivates the consideration of randomness as a design constraint.

From the prespective of the biologists attending this meeting, a simple fact will reveal the importance of synthetic simulations. The dynamic lateral inhibition presented in figure 6 is known to exist in rat somatosensory cortex (Simons, 1985). By deflecting the whiskers on a rat's face, Simons was able to stimulate individual barrels of the postero-medial somatosensory barrel field in combinations which revealed similar spatio-temporal interactions among the responses of the cortical neurons of the barrel field. The temporal suppression he reported even has a time course similar to that of the simulation. What the experiment did not reveal, however, was the class of cell in which suppression was seen; the simulation located most of the suppression in the output pyramidal cells. Hence, for a biologist, even a simple synthetic simulation like the one presented here can make definitive predictions. What differentiates the predictions made by synthetic simulations from those of more general artificial neural systems, of course, is that the strong biological foundations of synthetic simulations provide an easily grasped and highly relevant framework for both predictions and experimental verification.

One of the advertised purposes of this meeting was to "bring together neurobiologists, cognitive psychologists, engineers, and physicists with common interest in natural and artificial neural networks." Towards that end, synthetic computer simulations, i.e. simulations which follow known neurophysiological and neuroanatomical data as if they comprised a complex recipe, can provide an experimental medium which is useful for both biologists and engineers. The simulation of cerebral neocortex developed here has information regarding the role of randomness in the the robustness and presence of various computational capabilities as well as information regarding the value of distinct levels of connectivity to contribute to the design of artificial neural networks. At the same time, the synthetic nature of the network provides the biologist with an environment in which he can test notions of actual neural function as well as with a system which replicates known properties of biological systems and makes explicit predictions. Providing two-way interactions, synthetic simulations like this one will allow future generations of artificial neural networks to benefit from the empirical findings of biologists, while the slowly evolving theories of brain function benefit from the more generalizable results and methods of engineers.

## Footnotes

* Because neurons receive varying amounts of input and because integration is performed by summating excitatory and inhibitory postsynaptic potentials in a nonlinear way, the time each neuron needs to summate its inputs and produce an action potential varies from neuron to neuron and from time to time. This then allows for asynchronous firing in spite of the identical transmission times.

## References

Crick, F. H. C. (1979) Thinking about the brain, *Scientific American*, **241**:219 - 232.

Houchin, J. (1975) Direction specificity in cortical responses to moving stimuli -- a simple model. *Proceedings of the Physiological Society*, **247**:7 - 9.

Jones, E. G. (1986) Connectivity of primate sensory-motor cortex , in *Cerebral Cortex*, vol. 5, E. G. Jones and A. Peters (eds), Plenum Press, New York.

Koch, C., Poggio, T., and Torre, V. (1983) Nonlinear interactions in a dendritic tree: Localization, timing, and role in information processing. *Proceedings of the National Academy of Science, USA*, **80**:2799 - 2802.

Lewin, R. (1982) Neuroscientists look for theories, *Science*, **216**:507.

MacGregor, R.J. and Lewis, E.R. (1977) *Neural Modeling*, Plenum Press, New York.

Mountcastle, V. B. (1978) An organizing principle for cerebral function: The unit module and the distributed system, in *The Mindful Brain*, G. M. Edelman and V. B. Mountcastle (eds.), MIT Press, Cambridge, MA.

Shepherd, G.M. (in press) Basic circuit of cortical organization, in *Perspectives in Memory Research*, M.S. Gazzaniga (ed.), MIT Press, Cambridge, MA.
Simons, D. J. (1985) Temporal and spatial integration in the rat SI vibrissa cortex, *Journal of Neurophysiology*, **54**:615 - 635.

Szenthágothai, J. (1978) Specificity versus (quasi-) randomness in cortical connectivity, in *Architectonics of the Cerebral Cortex*, M. A. B. Brazier and H. Petsche (eds.), Raven Press, New York.

Szentágothai, J. (1979) Local neuron circuits in the neocortex, in *The Neurosciences. Fourth Study Program*, F. O. Schmitt and F. G. Worden (eds.), MIT Press, Cambridge, MA.

Torre, V. and Poggio, T. (1978) A synaptic mechanism possibly underlying directional selectivity to motion, *Proceeding of the Royal Society (London) B*, **202**:409 -416.

Woolsey, T.A. and Van der Loos, H. (1970) Structural organization of layer IV in the somatosensory region (SI) of mouse cerebral cortex, *Brain Research*, **17**:205-242.

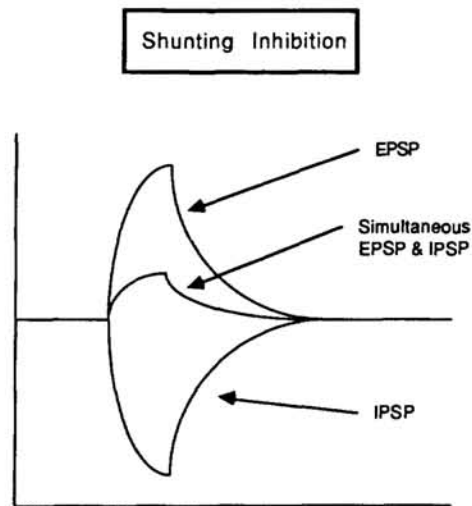

**Figure 1A:** Intracellular records of post-synaptic potentials resulting from single excitatory and inhibitory inputs to cells at different resting potentials.

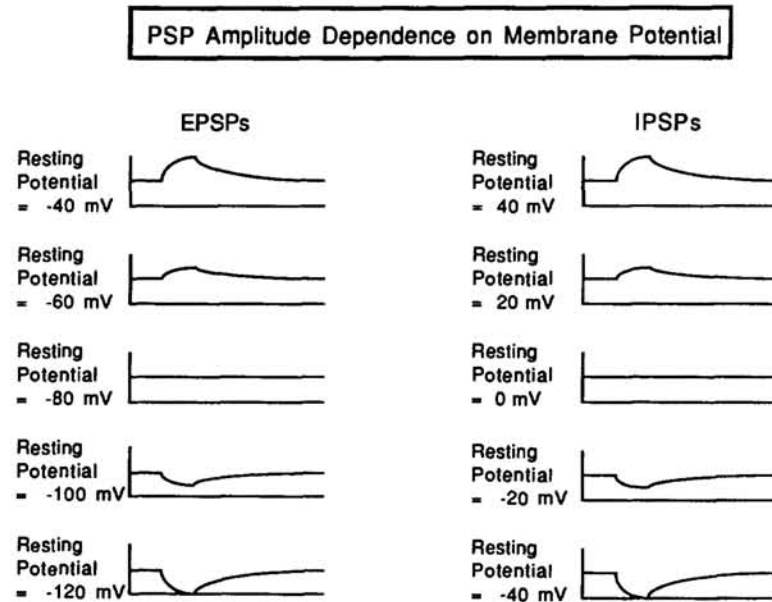

**Figure 1B:** Illustration of the current shunting nonlinearity present in the model neurons. Though the simultaneous arrival of postsynaptic potentials of equal and opposite amplitude would result in no deflection in the membrane potential of a simple linear neuron model, a variety of factors contribute to the nonlinear response of actual neurons and of the neurons modeled in the present simulation.

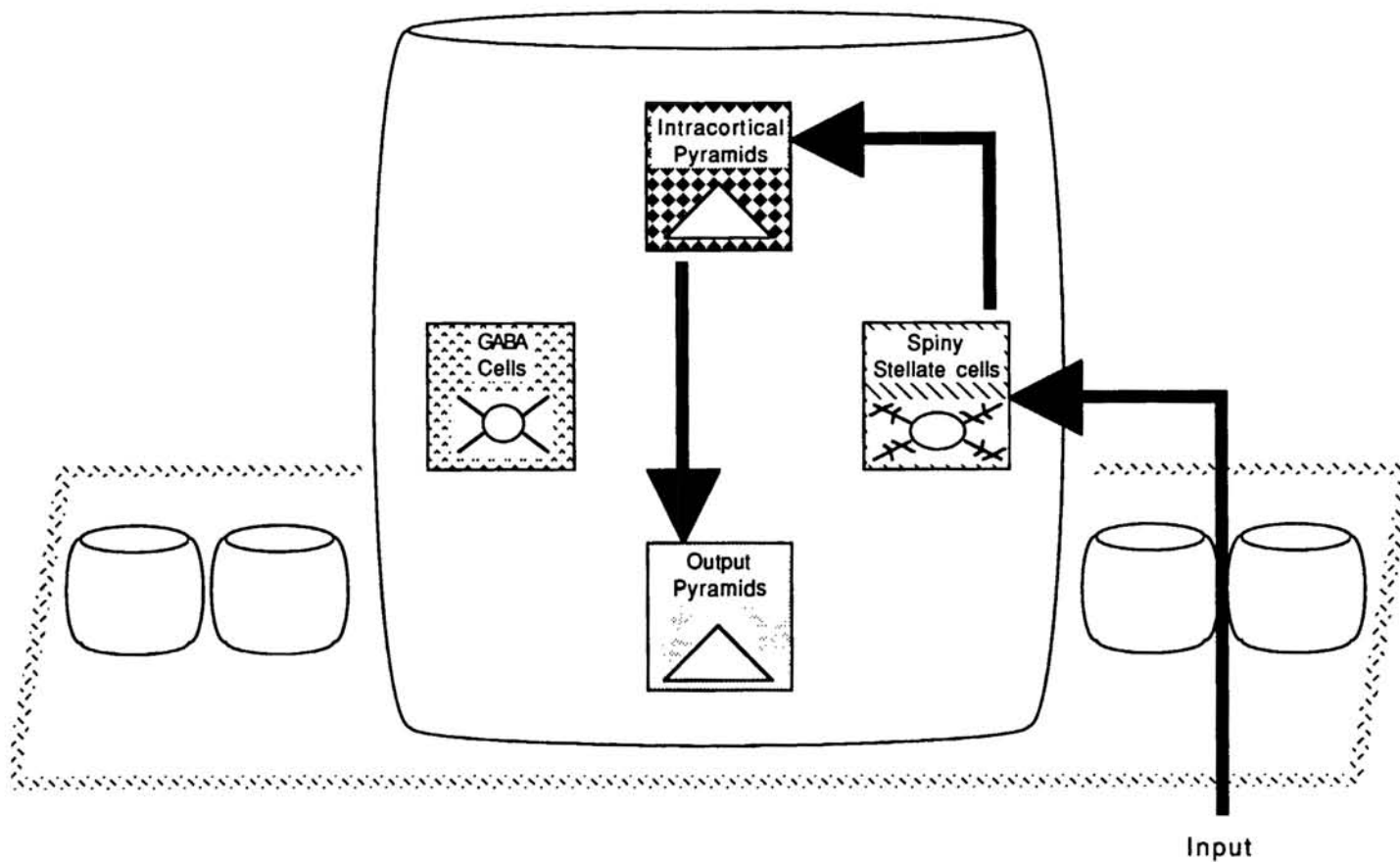

**Figure 2:** A schematic representation of the simulated cortical network. Five modules are used, each containing sixty neurons. Neurons are divided into four classes. Numerals within the caricatured neurons represent the number of cells in that particular class that are simulated. Though all cell classes are connected to all other classes, the pathway from input to spiny stellate to cortico-cortical pyramids to output pyramids is particularly strong.

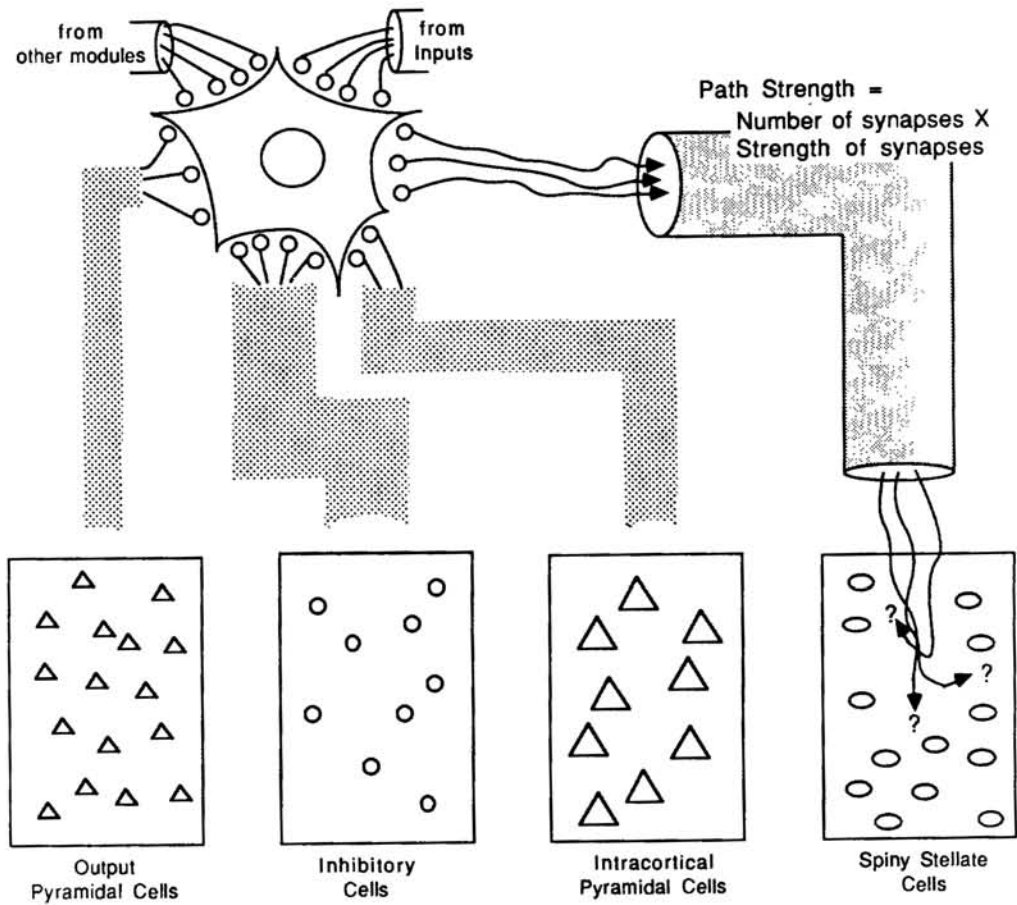

**Figure 3:** Two levels of connectivity are defined in the network. Gross synaptic architecture is defined among classes of cells. Fine-grain connectivity specifies which cell connects to which other cell and is determined at random.

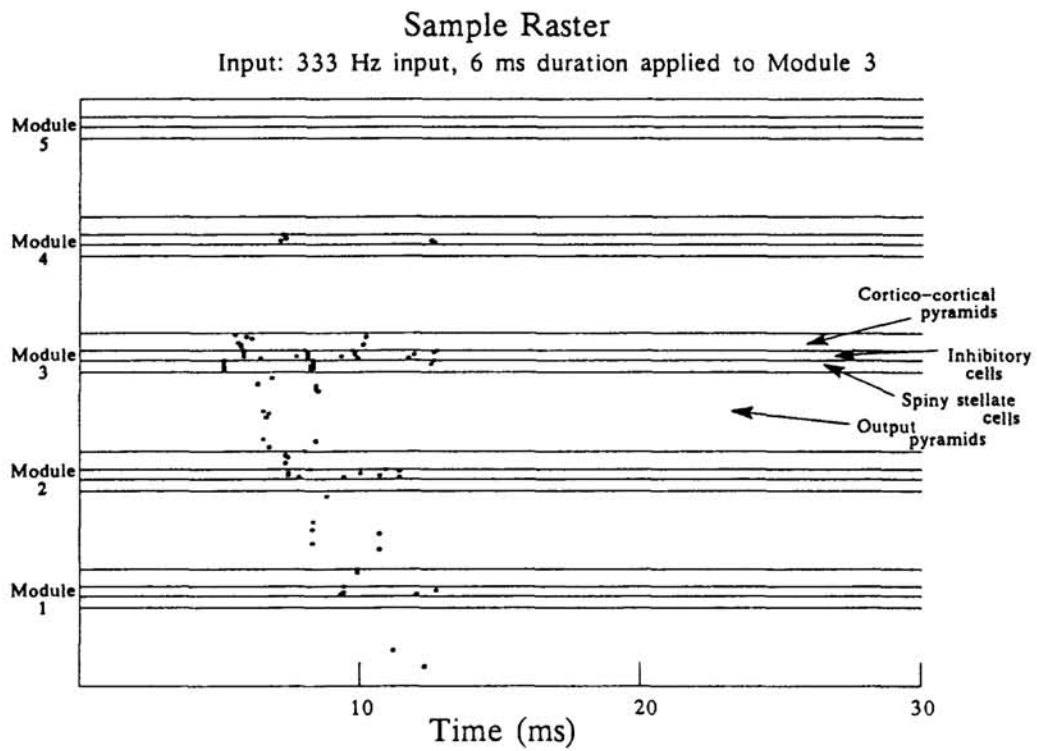

**Figure 4:** Sample response of the entire network to a small burst of action potentials delivered to module 3.

## Robustness With Respect to Connectivity Pattern

### Synaptic Architecture Constant

**Figure 5:** Plot of an arbitrary activity measure (total spike activity in all pyramidal cells) versus various instatiations of the same connectional architecture. Along the abscissa are represented the different fine-grained patterns of connectivity within a fixed connectional architecture. In one case the conductance times between all cells was 1 msec and in the other case the times were selected at random from values between 0.1 msec and 2 msec. This experiment shows the greater overall stability produced by random conduction times.

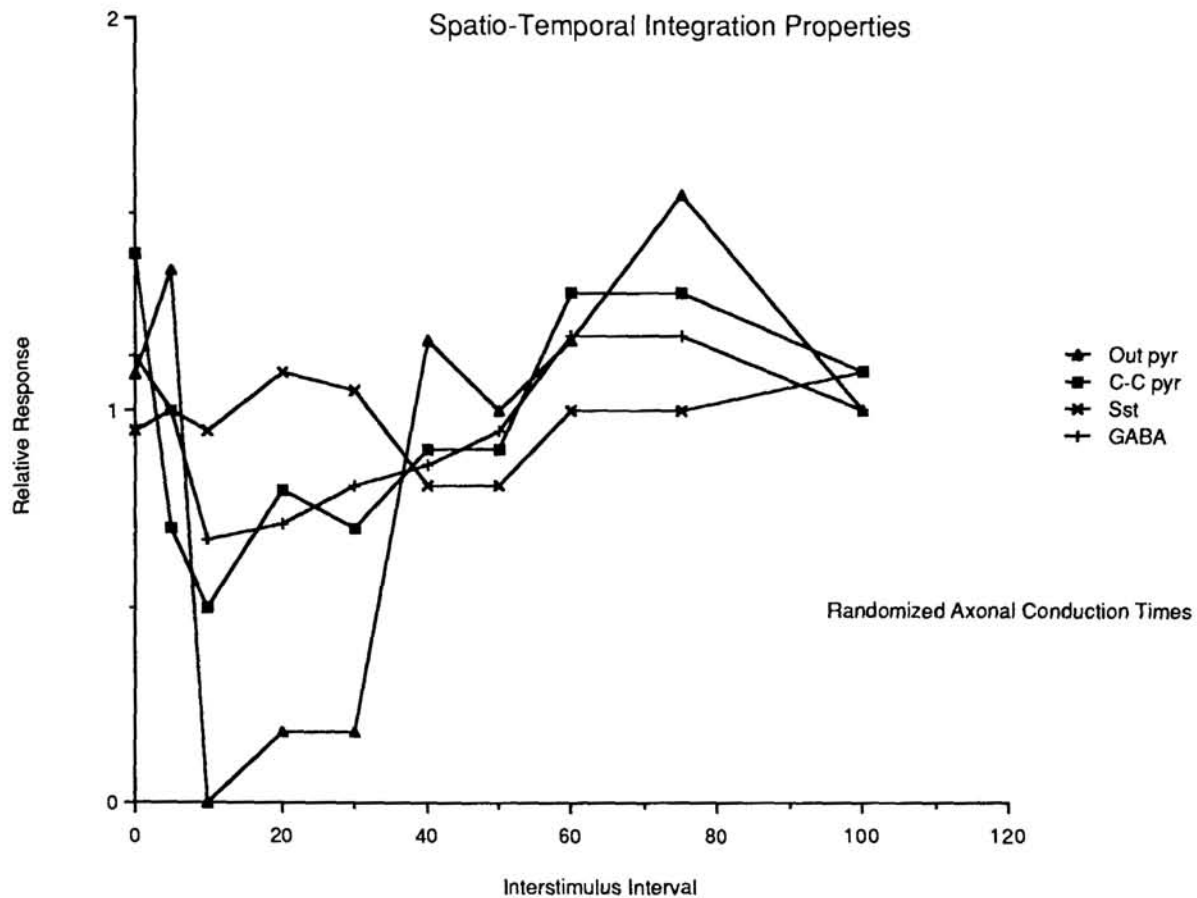

**Figure 6:** Spatio-temporal integration within the network. Plot of the time course of response suppression in the various cell classes. The ordinate plots the ratio of average cell activity (in terms of spikes) to a direct input after the presentation of an input to a neighboring module, and the average reponse to an input in the absence of prior input to an adjacent module. Values greater than one represent an enhancement of activity in response to the spatially distinct preceeding input, while values less than one represent a suppression of the normal reponse. The abscissa plots the interstimulus interval. Note that the response suppression is most striking in only one class of cells.

